# Ensemble and Modular Approaches for Face Detection: a Comparison

Raphaël Feraud *and Olivier Bernier [†]
France-Télécom CNET DTL/DLI
Technopole Anticipa, 2 avenue Pierre Marzin, 22307 Lannion cedex, FRANCE

## Abstract

A new learning model based on autoassociative neural networks is developped and applied to face detection. To extend the detection ability in orientation and to decrease the number of false alarms, different combinations of networks are tested: ensemble, conditional ensemble and conditional mixture of networks. The use of a conditional mixture of networks allows to obtain *state of the art* results on different benchmark face databases.

## 1 A constrained generative model

Our purpose is to classify an extracted window $x$ from an image as a face ($x \in \mathcal{V}$) or non-face ($x \in \mathcal{N}$). The set of all possible windows is $\mathcal{E} = \mathcal{V} \cup \mathcal{N}$, with $\mathcal{V} \cap \mathcal{N} = \emptyset$. Since collecting a representative set of non-face examples is impossible, face detection by a statistical model is a difficult task. An autoassociative network, using five layers of neurons, is able to perform a non-linear dimensionnality reduction [Kramer, 1991]. However, its use as an estimator, to classify an extracted window as face or non-face, raises two problems:

1. $\mathcal{V}'$, the obtained sub-manifold can contain non-face examples ($\mathcal{V} \subset \mathcal{V}'$),

2. owing to local minima, the obtained solution can be close to the linear solution: the principal components analysis.

Our approach is to use counter-examples in order to find a sub-manifold as close as possible to $\mathcal{V}$ and to constrain the algorithm to converge to a non-linear solution [Feraud, R. et al., 1997]. Each non-face example is constrained to be reconstructed as its projection on $\mathcal{V}$. The projection $\mathcal{P}$ of a point $x$ of the input space $\mathcal{E}$ on $\mathcal{V}$, is defined by:

[†]email: bernier@lannion.cnet.fr

- if $x \in \mathcal{V}$, then $\mathcal{P}(x) = x$,
- if $x \notin \mathcal{V}$: $\mathcal{P}(x) = \arg\min_{y \in \mathcal{V}}(d(x, y))$, where $d$ is the Euclidian distance. During the learning process, the projection $\mathcal{P}$ of $x$ on $\mathcal{V}$ is approximated by: $\mathcal{P}(x) \sim \frac{1}{n}\sum_{i=1}^{n} v_i$, where $v_1, v_2, \ldots, v_n$, are the $n$ nearest neighbours, in the training set of faces, of $v$, the nearest face example of $x$.

The goal of the learning process is to approximate the distance $\mathcal{D}$ of an input space element $x$ to the set of faces $\mathcal{V}$:

- $\mathcal{D}(x, \mathcal{V}) = \|x - \mathcal{P}(x)\| \sim \frac{1}{M}(x - \hat{x})^2$, where $M$ is the size of input image $x$ and $\hat{x}$ the image reconstructed by the neural network,
- let $x \in \mathcal{E}$, then $x \in \mathcal{V}$ if and only if $\mathcal{D}(x, \mathcal{V}) \leq \tau$, with $\tau \in \mathbb{R}$, where $\tau$ is a threshold used to adjust the sensitivity of the model.

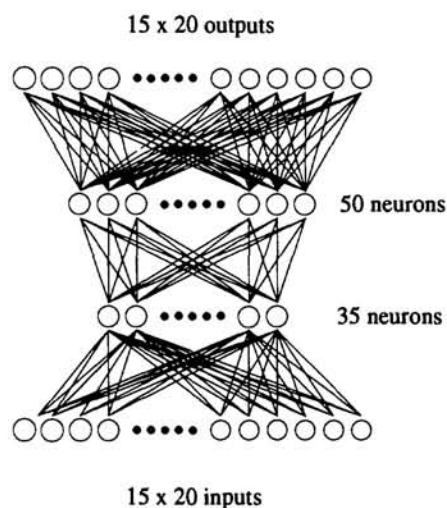

15 x 20 outputs

50 neurons

35 neurons

15 x 20 inputs

Figure 1: The use of two hidden layers and counter-examples in a compression neural network allows to realize a non-linear dimensionality reduction.

In the case of non-linear dimensionnality reduction, the reconstruction error is related to the position of a point to the non-linear principal components in the input space. Nevertheless, a point can be near to a principal component and far from the set of faces. With the algorithm proposed, the reconstruction error is related to the distance between a point to the set of faces. As a consequence, if we assume that the learning process is consistent [Vapnik, 1995], our algorithm is able to evaluate the probability that a point belongs to the set of faces. Let $y$ be a binary random variable: $y = 1$ corresponds to a face example and $y = 0$ to a non-face example, we use:

$$P(y = 1|x) = e^{-\frac{(x - \hat{x})^2}{\sigma^2}} \text{, where } \sigma \text{ depends on the threshold } \tau$$

The size of the training windows is 15x20 pixels. The faces are normalized in position and scale. The windows are enhanced by histogram equalization to obtain a relative independence to lighting conditions, smoothed to remove the noise and normalized by the average face, evaluated on the training set. Three face databases are used: after vertical mirroring, $B_{f1}$ is composed of 3600 different faces with orientation between 0 degree and 20 degree, $B_{f2}$ is composed of 1600 different faces with orientation between 20 degree and 60 degree and $B_{f3}$ is the concatenation of $B_{f1}$ and $B_{f2}$, giving a total of 5200 faces. All of the training faces are extracted

from the *usenix face database*(\*\*), from the test set B of CMU(\*\*), and from 100 images containing faces and complex backgrounds.

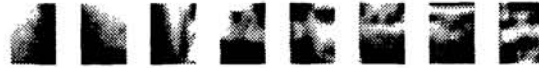

Figure 2: Left to right: the counter-examples successively chosen by the algorithm are increasingly similar to real faces (iteration 1 to 8).

The non-face databases $(B_{nf1}, B_{nf2}, B_{nf3})$, corresponding to each face database, are collected by an iterative algorithm similar to the one used in [Sung, K. and Poggio, T., 1994] or in [Rowley, H. et al., 1995]:

- **1)** $B_{nf} = \emptyset$, $\tau = \tau_{min}$,
- **2)** the neural network is trained with $B_f + B_{nf}$,
- **3)** the face detection system is tested on a set of background images,
- **4)** a maximum of 100 subimages $x_i$ are collected with $\mathcal{D}\,(x_i, \mathcal{V}) \leq \tau$,
- **5)** $B_{nf} = B_{nf} + \{x_0, \ldots, x_n\}$, $\tau = \tau + \mu$, with $\mu > 0$,
- **6)** while $\tau < \tau_{max}$ go back to step 2.

After vertical mirroring, the size of the obtained set of non-face examples is respectively 1500 for $B_{nf1}$, 600 for $B_{nf2}$ and 2600 for $B_{nf3}$. Since the non-face set $(\mathcal{N})$ is too large, it is not possible to prove that this algorithm converge in a finite time. Nevertheless, in only 8 iterations, collected counter-examples are close to the set of faces (Figure 2). Using this algorithm, three elementary face detectors are constructed: the front view face detector trained on $B_{f1}$ and $B_{nf1}$ (CGM1), the turned face detector trained on $B_{f2}$ and $B_{nf2}$ (CGM2) and the general face detector trained on $B_{f3}$ and $B_{nf3}$ (CGM3).

To obtain a non-linear dimensionnality reduction, five layers are necessary. However, our experiments show that four layers are sufficient. Consequently, each CGM has four layers (Figure 1). The first and last layers consist each of 300 neurons, corresponding to the image size 15x20. The first hidden layer has 35 neurons and the second hidden layer 50 neurons. In order to reduce the false alarm rate and to extend the face detection ability in orientation, different combinations of networks are tested. The use of ensemble of networks to reduce false alarm rate was shown by [Rowley, H. et al., 1995]. However, considering that to detect a face in an image, there are two subproblems to solve, detection of front view faces and turned faces, a modular architecture can also be used.

## 2   Ensemble of CGMs

Generalization error of an estimator can be decomposed in two terms: the bias and the variance [Geman, S. et al., 1992]. The bias is reduced with prior knowledge. The use of an ensemble of estimators can reduce the variance when these estimators are independently and identically distributed [Raviv, Y. and Intrator, N., 1996]. Each face detector $i$ produces:

$$E_i[y|x] = P_i(y = 1|x)$$

Assuming that the three face detectors (CGM1,CGM2,CGM3) are independently and identically distributed (iid), the ouput of the ensemble is:

$$E[y|x] = \frac{1}{3}\sum_{i=1}^{3} E_i[y|x]$$

# 3   Conditional mixture of CGMs

To extend the detection ability in orientation, a conditional mixture of CGMs is tested. The training set is separated in two subsets: front view faces and the corresponding counter-examples ($\theta = 1$) and turned faces and the corresponding counter-examples ($\theta = 0$). The first subnetwork (CGM1) evaluates the probability of the tested image to be a front view face, knowing the label equals 1 ($P(y = 1|x, \theta = 1)$). The second (CGM2) evaluates the probability of the tested image to be a turned face, knowing the label equals 0 ($P(y = 1|x, \theta = 0)$). A gating network is trained to evaluate $P(\theta = 1|x)$, supposing that the partition $\theta = 1, \theta = 0$ can be generalized to every input:

$$E[y|x] = E[y|\theta = 1, x]f(x) + E[y|\theta = 0, x](1 - f(x))$$

Where $f(x)$ is the estimated value of $P(\theta = 1|x)$.

This system is different from a mixture of experts introduced by [Jacobs, R. A. et al., 1991]: each module is trained separately on a subset of the training set and then the gating network learns to combine the outputs.

# 4   Conditional ensemble of CGMs

To reduce the false alarm rate and to detect front view and turned faces, an original combination, using (CGM1,CGM2) and a gate network, is proposed. Four sets are defined:

- $\mathcal{F}$ is the front view face set,
- $\mathcal{P}$ is the turned face set, with $\mathcal{F} \cap \mathcal{P} = \emptyset$,
- $\mathcal{V} = \mathcal{F} \cup \mathcal{P}$ is the face set,
- $\mathcal{N}$ is the non-face set, with $\mathcal{V} \cap \mathcal{N} = \emptyset$,

Our goal is to evaluate $P(x \in V|x)$. Each estimator computes respectively:

- $P(x \in F|x \in \mathcal{F} \cup \mathcal{N}, x)$ ($CGM1(x)$),
- $P(x \in P|x \in \mathcal{P} \cup \mathcal{N}, x)$ ($CGM2(x)$),

Using the Bayes theorem, we obtain:

$$P(x \in \mathcal{F}|x \in \mathcal{F} \cup \mathcal{N}, x) = \frac{P(x \in \mathcal{F}|x)}{P(x \in \mathcal{F} \cup \mathcal{N}|x)} P(x \in \mathcal{F} \cup \mathcal{N}|x \in \mathcal{F}, x)$$

Since $x \in \mathcal{F} \Rightarrow x \in \mathcal{F} \cup \mathcal{N}$, then:

$$P(x \in \mathcal{F}|x, x \in \mathcal{F} \cup \mathcal{N}) = \frac{P(x \in \mathcal{F}|x)}{P(x \in \mathcal{F} \cup \mathcal{N}|x)}$$

$$\Leftrightarrow P(x \in \mathcal{F}|x) = P(x \in \mathcal{F}|x \in \mathcal{F} \cup \mathcal{N}, x)P(x \in \mathcal{F} \cup \mathcal{N}|x)$$

$$\Leftrightarrow P(x \in \mathcal{F}|x) = P(x \in \mathcal{F}|x \in \mathcal{F} \cup \mathcal{N}, x)[P(x \in \mathcal{F}|x) + P(x \in \mathcal{N}|x)]$$

In the same way, we have:

$$P(x \in \mathcal{P}|x) = P(x \in \mathcal{P}|x \in \mathcal{P} \cup \mathcal{N}, x)[P(x \in \mathcal{P}|x) + P(x \in \mathcal{N}|x)]$$

Then:

$$P(x \in \mathcal{V}|x) = P(x \in \mathcal{N}|x)[P(x \in \mathcal{P}|x \in \mathcal{P} \cup \mathcal{N}, x) + P(x \in \mathcal{F}|x \in \mathcal{F} \cup \mathcal{N}, x)]$$
$$+ P(x \in \mathcal{P}|x)P(x \in \mathcal{P}|x \in \mathcal{P} \cup \mathcal{N}, x) + P(x \in \mathcal{F}|x)P(x \in \mathcal{F}|x \in \mathcal{F} \cup \mathcal{N}, x)$$

Rewriting the previous equation using the following notation, $CGM1(x)$ for $P(x \in \mathcal{F}|x \in \mathcal{F} \cup \mathcal{N}, x)$ and $CGM1(x)$ for $P(x \in \mathcal{P}|x \in \mathcal{P} \cup \mathcal{N}, x)$, we have:

$$P(x \in \mathcal{V}|x) = P(x \in \mathcal{N}|x)[CGM1(x) + CGM2(x)] \ (1)$$
$$+ P(x \in \mathcal{P}|x)CGM2(x) + P(x \in \mathcal{F}|x)CGM1(x) \ (2)$$

Then, we can deduce the behaviour of the conditional ensemble:

- in $\mathcal{N}$, if the output of the gate network is 0.5, as in the case of ensembles, the conditional ensemble reduces the variance of the error (first term of the right side of the equation (1)),
- in $\mathcal{V}$, as in the case of the conditional mixture, the conditional ensemble permits to combine two different tasks (second term of the right side of the equation (2)): detection of turned faces and detection of front view faces.

The gate network $f(x)$ is trained to calculate the probability that the tested image is a face ($P(x \in \mathcal{V}|x)$), using the following cost function:

$$C = \sum_{x_i \in \mathcal{V}} ([f(x_i)MGC1(x) + (1 - f(x_i))]MGC2(x) - y_i)^2 + \sum_{x_i \in \mathcal{N}} (f(x_i) - 0.5)^2$$

## 5   Discussion

Each 15x20 subimage is extracted and normalized by enhancing, smoothing and substracting the average face, before being processed by the network. The detection threshold $\tau$ is fixed for all the tested images. To detect a face at different scales, the image is subsampled.

The first test allows to evaluate the limits in orientation of the face detectors. The *sussex face database*(**), containing different faces with ten orientations betwen 0 degree and 90 degrees, is used (Table 1). The general face detector (CGM3) uses the same learning face database than the different mixtures of CGMs. Nevertheless, CGM3 has a smaller orientation range than the conditional mixtures of CGMs, and the conditional ensemble of CGMs. Since the performances of the ensemble of CGMS are low, the corresponding hypothesis (the CGMs are iid) is invalid. Moreover, this test shows that the combination by a gating neural network of CGMs,

Table 1:Results on *Sussex face database*

| orientation (degree) | CGM1 | CGM2 | CGM3 | Ensemble (1,2,3) | Conditional ensemble (1,2,gate) | Conditional mixture (1,2,gate) |
|---|---|---|---|---|---|---|
| 0 | 100.0 % | 100.0 % | 100.0 % | 100.0 % | 100.0 % | 100.0 % |
| 10 | 62.5 % | 100.0 % | 87.5 % | 100.0 % | 100.0 % | 100.0 % |
| 20 | 50.0 % | 100.0 % | 87.5 % | 87.5 % | 100.0 % | 100.0 % |
| 30 | 12.5 % | 100.0 % | 62.5 % | 62.5 % | 100.0 % | 100.0 % |
| 40 | 0.0 % | 100.0 % | 50.0 % | 12.5 % | 62.5.0 % | 87.5 % |
| 50 | 0.0 % | 75.0 % | 0.0 % | 0.0 % | 37.5 % | 62.5 % |
| 60 | 0.0 % | 37.5 % | 0.0 % | 0.0 % | 0.0 % | 37.5 % |
| 70 | 0.0 % | 37.5 % | 0.0 % | 0.0 % | 0.0 % | 25.0 % |

trained on different training set, allows to extend the detection ability to both front view and turned faces. The conditional mixture of CGMs obtains results in term of orientation and false alarm rate close to the best CGMs used to contruct it (see Table 1 and Table 2).

The second test allows to evaluate the false alarms rate and to compare our results with the best results published so far on the test set $A$ [Rowley, H. et al., 1995] of the CMU (**), containing 42 images of various quality. First, these results show that the model, trained without counter-examples (GM), overestimates the distribution of faces and its false alarm rate is too important to use it as a face detector. Second, the estimation of the probability distribution of the face performed by one CGM (CGM3) is more precise than the one obtained by [Rowley, H. et al., 1995] with one SWN (see Table 2). The conditional ensemble of CGMs and the conditional mixture of CGMs obtained a similar detection rate than an ensemble of SWNs [Rowley, H. et al., 1995], but with a false alarm rate two or three times lower. Since the results of the conditional ensemble of CGMs and the conditional mixture of CGMs are close on this test, the detection rate versus the number of false alarms is plotted (Figure 3), for different thresholds. The conditional mixture of CGMs curve is above the one for the conditional ensemble of CGMs.

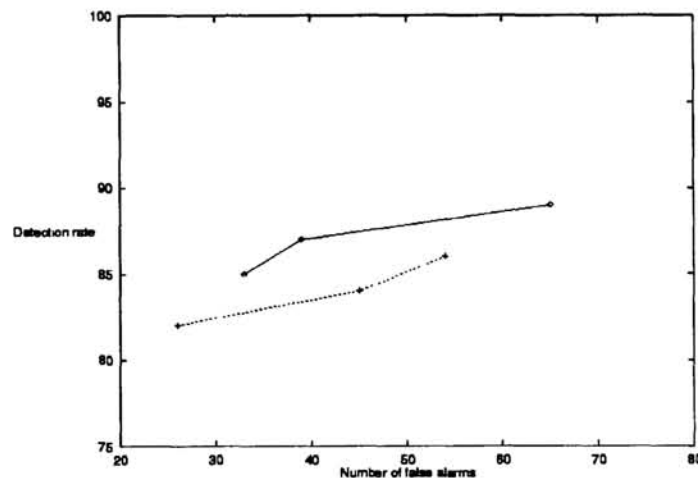

Figure 3: Detection rate versus number of false alarms on the CMU test set A. In dashed line conditional ensemble and in solid line conditional mixture.

The conditionnal mixture of CGMs is used in an application called LISTEN [Collobert, M. et al., 1996]: a camera detects, tracks a face and controls a microphone array towards a speaker, in real time. The small size of the subimages (15x20) processed allows to detect a face far from the camera (with an aperture of 60 degrees, the maximum distance to the camera is 6 meters). To detect a face in real time, the number of tested hypothesis is reduced by motion and color analysis.

Table 2:Results on the CMU test set A
GM: the model trained without counter-examples, CGM1: face detector, CGM2:
turned face detector, CGM3: general face detector. SWN: shared weight network.
(*) Considering that our goal is to detect human faces, non-human faces and rough
face drawings have not been taken into account.

| Model | Detection rate | | False alarms rate | |
|---|---|---|---|---|
| GM | 84 % | +/- 5 % | 1/1,000 | +/- 0.1/1,000,000 |
| CGM1 | 77 % 127/164 | +/- 5 % | 5.43/1,000,000 47/33,700,000 | +/- 0.38/1,000,00 |
| CGM2 | 85 % 139/164 | +/- 5 % | 6.3/1,000,000 212/33,700,000 | +/- 0.37/1,000,00 |
| CGM3 | 85 % 139/164 | +/- 5 % | 1.36/1,000,000 46/33,700,000 | +/- 0.41/1,000,00 |
| [Rowley,1995] (one SWN) | 84 % 142/169* | +/- 5 % | 8.13/1,000,000 179/22,000,000 | +/- 0.4/1,000,000 |
| Ensemble (CGM1,CGM2,CGM3) | 74 % 121/164 | +/- 5 % | 0.71/1,000,000 24/33,700,000 | +/- 0.43/1,000,00 |
| Conditional ensemble (CGM1,CGM2,gate) | 82 % 134/164 | +/- 5 % | 0.77/1,000,000 26/33,700,000 | +/- 0.38/1,000,00 |
| [Rowley,1995] (three SWNs) | 85 % 144/169* | +/- 5 % | 2.13/1,000,000 47/22,000,000 | +/- 0.42/1,000,00 |
| Conditional mixture (CGM1,CGM2,gate) | 87 % 142/164 | +/- 5 % | 1.15/1,000,000 39/33,700,000 | +/- 0.35/1,000,00 |

(**) *usenix face database, sussex face database* and CMU test sets can be retrieved at
www.cs.rug.nl/ peterkr/FACE/face.html.

## Footnotes

*email: feraud@lannion.cnet.fr

# References

[Collobert, M. et al., 1996] Collobert, M., Feraud, R., Le Tourneur, G., Bernier, O., Viallet, J.E, Mahieux, Y., and Collobert, D. (1996). Listen: a system for locating and tracking individual speaker. In *Second International Conference On Automatic Face and Gesture Recognition*.

[Feraud, R. et al., 1997] Feraud, R., Bernier, O., and Collobert, D. (1997). A constrained generative model applied to face detection. *Neural Processing Letters*.

[Geman, S. et al., 1992] Geman, S., Bienenstock, E., and Doursat, R. (1992). Neural networks and the bias-variance dilemma. *Neural Computation*, 4:1–58.

[Jacobs, R. A. et al., 1991] Jacobs, R. A., Jordan, M. I., Nowlan, S. J., and Hinton, G. E. (1991). Adaptative mixtures of local experts. *Neural Computation*, 3:79–87.

[Kramer, 1991] Kramer, M. (1991). Nonlinear principal component analysis using autoassociative neural networks. *AIChE Journal*, 37:233–243.

[Raviv, Y. and Intrator, N., 1996] Raviv, Y. and Intrator, N. (1996). Bootstrapping with noise: An effective regularization technique. *Connection Science*, 8:355–372.

[Rowley, H. et al., 1995] Rowley, H., Baluja, S., and Kanade, T. (1995). Human face detection in visual scenes. In *Neural Information Processing Systems 8*.

[Sung, K. and Poggio, T., 1994] Sung, K. and Poggio, T. (1994). Example-based learning for view-based human face detection. Technical report, M.I.T.

[Vapnik, 1995] Vapnik, V. (1995). *The Nature of Statistical Learning Theory*. Springer-Verlag New York Heidelberg Berlin.
